# Multidimensional Triangulation and Interpolation for Reinforcement Learning

**Scott Davies**

scottd@cs.cmu.edu

Department of Computer Science, Carnegie Mellon University

5000 Forbes Ave, Pittsburgh, PA 15213

**Abstract**

Dynamic Programming, Q-learning and other discrete Markov Decision Process solvers can be applied to continuous $d$-dimensional state-spaces by quantizing the state space into an array of boxes. This is often problematic above two dimensions: a coarse quantization can lead to poor policies, and fine quantization is too expensive. Possible solutions are variable-resolution discretization, or function approximation by neural nets. A third option, which has been little studied in the reinforcement learning literature, is interpolation on a coarse grid. In this paper we study interpolation techniques that can result in vast improvements in the online behavior of the resulting control systems: multilinear interpolation, and an interpolation algorithm based on an interesting regular triangulation of $d$-dimensional space. We adapt these interpolators under three reinforcement learning paradigms: (i) offline value iteration with a known model, (ii) Q-learning, and (iii) online value iteration with a previously unknown model learned from data. We describe empirical results, and the resulting implications for practical learning of continuous non-linear dynamic control.

## 1 GRID-BASED INTERPOLATION TECHNIQUES

Reinforcement learning algorithms generate functions that map states to "cost-to-go" values. When dealing with continuous state spaces these functions must be approximated. The following approximators are frequently used:

- **Fine grids** may be used in one or two dimensions. Above two dimensions, fine grids are too expensive. Value functions can be discontinuous, which (as we will see) can lead to suboptimalities even with very fine discretization in two dimensions.

- **Neural nets** have been used in conjunction with TD [Sutton, 1988] and Q-learning [Watkins, 1989] in very high dimensional spaces [Tesauro, 1991, Crites and Barto, 1996]. While promising, it is not always clear that they produce the accurate value functions that might be needed for fine near-optimal control of dynamic systems, and the most commonly used methods of applying value iteration or policy iteration with a neural-net value function are often unstable. [Boyan and Moore, 1995].

Interpolation over points on a coarse grid is another potentially useful approximator for value functions that has been little studied for reinforcement learning. This paper attempts to rectify this omission. Interpolation schemes may be particularly attractive because they are local *averagers*, and convergence has been proven in such cases for offline value iteration [Gordon, 1995].

All of the interpolation methods discussed here split the state space into a regular grid of $d$-dimensional boxes; data points are associated with the centers or the corners of the resulting boxes. The value at a given point in the continuous state space is computed as a weighted average of neighboring data points.

## 1.1   MULTILINEAR INTERPOLATION

When using multilinear interpolation, data points are situated at the corners of the grid's boxes. The interpolated value within a box is an appropriately weighted average of the $2^d$ datapoints on that box's corners. The weighting scheme assures global continuity of the interpolated surface, and also guarantees that the interpolated value at any grid corner matches the given value of that corner.

In one-dimensional space, multilinear interpolation simply involves piecewise linear interpolations between the data points. In a higher-dimensional space, a recursive (though not terribly efficient) implementation can be described as follows:

- Pick an arbitrary axis. Project the query point along this axis to each of the two opposite faces of the box containing the query point.

- Use two $(d-1)$-dimensional multilinear interpolations over the $2^{d-1}$ datapoints on each of these two faces to calculate the values at both of these projected points.

- Linearly interpolate between the two values generated in the previous step.

Multilinear interpolation processes $2^d$ data points for every query, which becomes prohibitively expensive as $d$ increases.

## 1.2   SIMPLEX-BASED INTERPOLATION

It is possible to interpolate over $d+1$ of the data points for any given query in only $O(d \log d)$ time and still achieve a continuous surface that fits the datapoints exactly. Each box is broken into $d!$ hyperdimensional triangles, or *simplexes*, according to the *Coxeter-Freudenthal-Kuhn* triangulation [Moore, 1992].

Assume that the box is the unit hypercube, with one corner at $(x_1, x_2, \ldots, x_d) = (0, 0, \ldots, 0)$, and the diagonally opposite corner at $(1, 1, \ldots, 1)$. Then, each simplex in the Kuhn triangulation corresponds to one possible permutation $p$ of $(1, 2, \ldots, d)$, and occupies the set of points satisfying the equation

$$0 \leq x_{p(1)} \leq x_{p(2)} \leq \ldots \leq x_{p(d)} \leq 1.$$

Triangulating each box into $d!$ simplexes in this manner generates a *conformal mesh*: any two elements with a $(d-1)$-dimensional surface in common have entire faces in common, which ensures continuity across element boundaries when interpolating.

We use the Kuhn triangulation for interpolation as follows:

- Translate and scale to a coordinate system in which the box containing the query point is the unit hypercube. Let the new coordinate of the query point be $(x'_1, \ldots, x'_d)$.

- Use a sorting algorithm to rank $x'_1$ through $x'_d$. This tells us the simplex of the Kuhn triangulation in which the query point lies.

- Express $(x'_1, \ldots, x'_d)$ as a convex combination of the coordinates of the relevant simplex's $(d + 1)$ corners.
- Use the coefficients determined in the previous step as the weights for a weighted sum of the data values stored at the corresponding corners.

At no point do we explicitly represent the $d!$ different simplexes. All of the above steps can be performed in $O(d)$ time except the second, which can be done in $O(d \log d)$ time using conventional sorting routines.

## 2 PROBLEM DOMAINS

**CAR ON HILL:** In the Hillcar domain, the goal is to park a car near the top of a one-dimensional hill. The hill is steep enough that the driver needs to back up in order to gather enough speed to get to the goal. The state space is two-dimensional (position,velocity). See [Moore and Atkeson, 1995] for further details, but note that our formulation is harder than the usual formulation in that the goal region is restricted to a narrow range of velocities around 0, and trials start at random states. The task is specified by a reward of -1 for any action taken outside the goal region, and 0 inside the goal. No discounting is used, and two actions are available: maximum thrust backwards, and maximum thrust forwards.

**ACROBOT:** The Acrobot is a two-link planar robot acting in the vertical plane under gravity with a weak actuator at its elbow joint joint. The shoulder is un-actuated. The goal is to raise the hand to at least one link's height above the unactuated pivot [Sutton, 1996]. The state space is four-dimensional: two angular positions and two angular velocities. Trials always start from a stationary position hanging straight down. This task is formulated in the same way as the car-on-the-hill. The only actions allowed are the two extreme elbow torques.

## 3 APPLYING INTERPOLATION: THREE CASES

### 3.1 CASE I: OFFLINE VALUE ITERATION WITH A KNOWN MODEL

First, we precalculate the effect of taking each possible action from each state corresponding to a datapoint in the grid. Then, as suggested in [Gordon, 1995], we use these calculations to derive a *completely discrete* MDP. Taking any action from any state in this MDP results in $c$ possible successor states, where $c$ is the number of datapoints used per interpolation. Without interpolation, $c$ is 1; with multilinear interpolation, $2^d$; with simplex-based interpolation, $d + 1$.

We calculate the optimal policy for this derived MDP offline using *value iteration* [Ross, 1983]; because the value iteration can be performed on a completely discrete MDP, the calculations are much less computationally expensive than they would have been with many other kinds of function approximators. The value iteration gives us values for the datapoints of our grid, which we may then use to interpolate the values at other states during online control.

#### 3.1.1 Hillcar Results: value iteration with known model

We tested the two interpolation methods on a variety of quantization levels by first performing value iteration offline, and then starting the car from 1000 random states and averaging the number of steps taken to the goal from those states. We also recorded the number of backups required before convergence, as well as the execution time required for the entire value iteration on a 85 MHz Sparc 5. See Figure 1 for the results. All steps-to-goal values are means with an expected error of 2 steps.

| Interpolation Method | Grid size | | | |
|---|---|---|---|---|
| | $11^2$ | $21^2$ | $51^2$ | $301^2$ |
| **None** | | | | |
| Steps to Goal: | 237 | 131 | 133 | 120 |
| Backups: | 2.42K | 15.4K | 156K | 14.3M |
| Time (sec): | 0.4 | 1.0 | 4.1 | 192 |
| **MultiLin** | | | | |
| Steps to Goal: | 134 | 116 | 108 | 107 |
| Backups: | 4.84K | 18.1K | 205K | 17.8M |
| Time (sec): | 0.6 | 1.3 | 7.1 | 405 |
| **Simplex** | | | | |
| Steps to Goal: | 134 | 118 | 109 | 107 |
| Backups: | 6.17K | 18.1K | 195K | 17.9M |
| Time (sec): | 0.5 | 1.2 | 5.7 | 328 |

Figure 1: Hillcar: value iteration with known model

| Interpolation Method | Grid size | | | | | | | |
|---|---|---|---|---|---|---|---|---|
| | $8^4$ | $9^4$ | $10^4$ | $11^4$ | $12^4$ | $13^4$ | $14^4$ | $15^4$ |
| **None** | | | | | | | | |
| Steps to Goal: | - | - | 44089 | - | 26952 | - | > 100000 | - |
| Backups: | - | - | 280K | - | 622K | - | 1.42M | - |
| Time (sec): | - | - | 15 | - | 30 | - | 53 | - |
| **MultiLin** | | | | | | | | |
| Steps to Goal: | 3340 | 2006 | 1136 | 3209 | 1300 | 1820 | 1518 | 1802 |
| Backups: | 233K | 1.01M | 730K | 2.01M | 2.03M | 3.74M | 4.45M | 6.78M |
| Time (sec): | 17 | 43 | 42 | 83 | 99 | 164 | 197 | 284 |
| **Simplex** | | | | | | | | |
| Steps to Goal: | 4700 | 8007 | 2953 | 3209 | 4663 | 2733 | 1742 | 9613 |
| Backups: | 196K | 1.16M | 590K | 2.28M | 1.62M | 4.03M | 3.65M | 6.73M |
| Time (sec): | 9 | 24 | 22 | 47 | 47 | 86 | 93 | 142 |

Figure 2: Acrobot: value iteration with known model

The interpolated functions require more backups for convergence, but this is amply compensated by dramatic improvement in the policy. Surprisingly, both interpolation methods provide improvements even at extremely high grid resolutions – the noninterpolated grid with 301 datapoints along each axis fared no better than the interpolated grids with only 21 datapoints along each axis(!).

### 3.1.2   Acrobot Results: value iteration with known model

We used the same value iteration algorithm in the acrobot domain. In this case our test trials always began from the same start state, but we ran tests for a larger set of grid sizes (Figure 2).

Grids with different resolutions place grid cell boundaries at different locations, and these boundary locations appear to be important in this problem — the performance varies unpredictably as the grid resolution changes. However, in all cases, interpolation was necessary to arrive at a satisfactory solution; without interpolation, the value iteration often failed to converge at all. With relatively coarse grids it may be that any trajectory to the goal passes through some grid box more than once, which would immediately spell disaster for any algorithm associating a constant value over that entire grid box.

Controllers using multilinear interpolation consistently fared better than those employing the simplex-based interpolation; the smoother value function provided by multilinear interpolation seems to help. However, value iteration with the simplex-based interpolation was about twice as fast as that with multilinear interpolation. In higher dimensions this speed ratio will increase.

## 3.2 CASE II: Q-LEARNING

Under a second reinforcement learning paradigm, we do not use any model. Rather, we learn a *Q-function* that directly maps state-action pairs to long-term rewards [Watkins, 1989]. Does interpolation help here too?

In this implementation we encourage exploration by optimistically initializing the Q-function to zero everywhere. After travelling a sufficient distance from our last decision point, we perform a single backup by changing the grid point values according to a perceptron-like update rule, and then we greedily select the action for which the interpolated Q-function is highest at the current state.

### 3.2.1 Hillcar Results: Q-Learning

We used Q-Learning with a grid size of $11^2$. Figure 3 shows learning curves for three learners using the three different interpolation techniques.

Both interpolation methods provided a significant improvement in both initial and final online performance. The learner without interpolation achieved a final average performance of about 175 steps to the goal; with multilinear interpolation, 119; with simplex-based interpolation, 122. Note that these are all significant improvements over the corresponding results for offline value iteration with a known model. Inaccuracies in the interpolated functions often cause controllers to enter cycles; because the Q-learning backups are being performed online, however, the Q-learning controller can escape from these control cycles by depressing the Q-values in the vicinities of such cycles.

### 3.2.2 Acrobot Results: Q-Learning

We used the same algorithms on the acrobot domain with a grid size of $15^4$; results are shown in Figure 3.

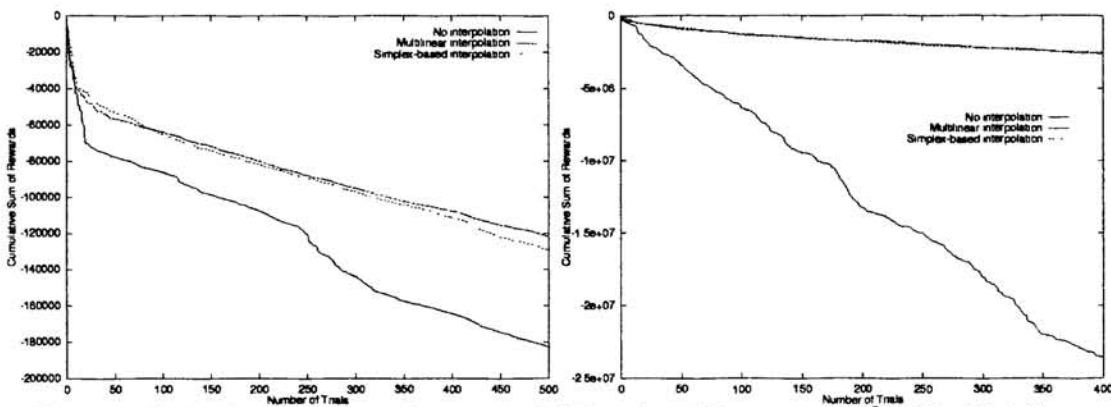

Figure 3: Left: Cumulative performance of Q-learning hillcar on an $11^2$ grid. (Multilinear interpolation comes out on top; no interpolation on the bottom.) Right: Q-learning acrobot on a $15^4$ grid. (The two interpolations come out on top with nearly identical performance.) For each learner, the y-axis shows the sum of rewards for all trials to date.
  The better the average performance, the shallower the gradient. Gradients are always negative because each state transition before reaching the goal results in a reward of -1.

Both Q-learners using interpolation improved rapidly, and eventually reached the goal in a relatively small number of steps per trial. The learner using multilinear interpolation eventually achieved an average of 1,529 steps to the goal per trial; the learner using simplex-based interpolation achieved 1,727 steps per trial. On the other hand, the learner not using any interpolation fared much worse, taking

an average of more than 27,000 steps per trial. (A controller that chooses actions randomly typically takes about the same number of steps to reach the goal.)

Simplex-based interpolation provided on-line performance very close to that provided by multilinear interpolation, but at roughly half the computational cost.

## 3.3 CASE III: VALUE ITERATION WITH MODEL LEARNING

Here, we use a model of the system, but we do not assume that we have one to start with. Instead, we learn a model of the system as we interact with it; we assume this model is adequate and calculate a value function via the same algorithms we would use if we knew the true model. This approach may be particularly beneficial for tasks in which data is expensive and computation is cheap. Here, models are learned using very simple grid-based function approximators without interpolation for both the reward and transition functions of the model. The same grid resolution is used for the value function grid and the model approximator. We strongly encourage exploration by initializing the model so that every state is initially assumed to be an absorbing state with zero reward.

While making transitions through the state space, we update the model and use prioritized sweeping [Moore and Atkeson, 1993] to concentrate backups on relevant parts of the state space. We also occasionally stop to recalculate the effects of all actions under the updated model and then run value iteration to convergence. As this is fairly time-consuming, it is done rather rarely; we rely on the updates performed by prioritized sweeping to guide the system in the meantime.

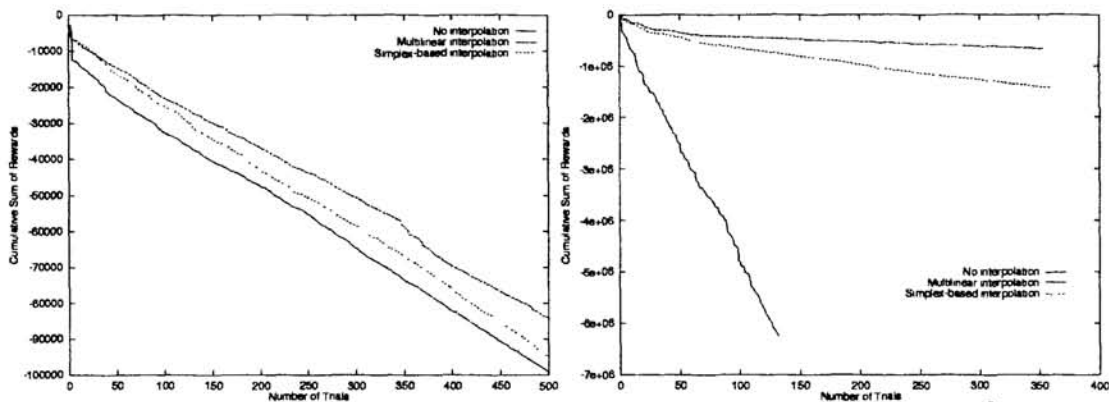

Figure 4: Left: Cumulative performance, model-learning on hillcar with a $11^2$ grid. Right: Acrobot with a $15^4$ grid. In both cases, multilinear interpolation comes out on top, while no interpolation winds up on the bottom.

### 3.3.1 Hillcar Results: value iteration with learned model

We used the algorithm described above with an 11-by-11 grid. An average of about two prioritized sweeping backups were performed per transition; the complete recalculations were performed every 1000 steps throughout the first two trials and every 5000 steps thereafter. Figure 4 shows the results for the first 500 trials.

Over the first 500 trials, the learner using simplex-based interpolation didn't fare much better than the learner using no interpolation. However, its performance on trials 1500-2500 (not shown) was close to that of the learner using multilinear interpolation, taking an average of 151 steps to the goal per trial while the learner using multilinear interpolation took 147. The learner using no interpolation did significantly worse than the others in these later trials, taking 175 steps per trial.

The model-learners' performance improved more quickly than the Q-learners' over the first few trials; on the other hand, their final performance was significantly worse that the Q-learners'.

### 3.3.2 Acrobot Results: value iteration with learned model
We used the same algorithm with a $15^4$ grid on the acrobot domain, this time performing the complete recalculations every 10000 steps through the first two trials and every 50000 thereafter. Figure 4 shows the results. In this case, the learner using no interpolation took so much time per trial that the experiment was aborted early; after 100 trials, it was still taking an average of more than 45,000 steps to reach the goal. The learners using interpolation, however, fared much better. The learner using multilinear interpolation converged to a solution taking 938 steps per trial; the learner using simplex-based interpolation averaged about 2450 steps. Again, as the graphs show, these three learners initially improve significantly faster than did the Q-Learners using similar grid sizes.

## 4 CONCLUSIONS
We have shown how two interpolation schemes—one based on a weighted average of the $2^d$ points in a square cell, the other on a $d$- dimensional triangulation—may be used in three reinforcement learning paradigms: Optimal policy computation with a known model, Q-learning, and online value iteration while learning a model. In each case our empirical studies demonstrate interpolation resoundingly decreasing the quantization level necessary for a satisfactory solution. Future extensions of this research will explore the use of variable resolution grids and triangulations, multiple low-dimensional interpolations in place of one high-dimension interpolation in a manner reminiscent of CMAC [Albus, 1981], memory-based approximators, and more intelligent exploration.

This research was funded in part by a National Science Foundation Graduate Fellowship to Scott Davies, and a Research Initiation Award to Andrew Moore.

## References

[Albus, 1981] J. S. Albus. *Brains, Behaviour and Robotics*. BYTE Books, McGraw-Hill, 1981.

[Boyan and Moore, 1995] J. A. Boyan and A. W. Moore. Generalization in Reinforcement Learning: Safely Approximating the Value Function. In *Neural Information Processing Systems 7*, 1995.

[Crites and Barto, 1996] R. H. Crites and A. G. Barto. Improving Elevator Performance using Reinforcement Learning. In D. Touretzky, M. Mozer, and M. Hasselmo, editors, *Neural Information Processing Systems 8*, 1996.

[Gordon, 1995] G. Gordon. Stable Function Approximation in Dynamic Programming. In *Proceedings of the 12th International Conference on Machine Learning*. Morgan Kaufmann, June 1995.

[Moore and Atkeson, 1993] A. W. Moore and C. G. Atkeson. Prioritized Sweeping: Reinforcement Learning with Less Data and Less Real Time. *Machine Learning*, 13, 1993.

[Moore and Atkeson, 1995] A. W. Moore and C. G. Atkeson. The Parti-game Algorithm for Variable Resolution Reinforcement Learning in Multidimensional State-spaces. *Machine Learning*, 21, 1995.

[Moore, 1992] D. W. Moore. Simplical Mesh Generation with Applications. PhD. Thesis. Report no. 92-1322, Cornell University, 1992.

[Ross, 1983] S. Ross. *Introduction to Stochastic Dynamic Programming*. Academic Press, New York, 1983.

[Sutton, 1988] R. S. Sutton. Learning to Predict by the Methods of Temporal Differences. *Machine Learning*, 3:9–44, 1988.

[Sutton, 1996] R. S. Sutton. Generalization in Reinforcement Learning: Successful Examples Using Sparse Coarse Coding. In D. Touretzky, M. Mozer, and M. Hasselmo, editors, *Neural Information Processing Systems 8*, 1996.

[Tesauro, 1991] G. J. Tesauro. Practical Issues in Temporal Difference Learning. RC 17223 (76307), IBM T. J. Watson Research Center, NY, 1991.

[Watkins, 1989] C. J. C. H. Watkins. Learning from Delayed Rewards. PhD. Thesis, King's College, University of Cambridge, May 1989.
